# A Model for Temporal Dependencies in Event Streams

**Asela Gunawardana**
Microsoft Research
One Microsoft Way
Redmond, WA 98052
aselag@microsoft.com

**Christopher Meek**
Microsoft Research
One Microsoft Way
Redmond, WA 98052
meek@microsoft.com

**Puyang Xu**
ECE Dept. & CLSP
Johns Hopkins University
Baltimore, MD 21218
puyangxu@jhu.edu

## Abstract

We introduce the Piecewise-Constant Conditional Intensity Model, a model for learning temporal dependencies in event streams. We describe a closed-form Bayesian approach to learning these models, and describe an importance sampling algorithm for forecasting future events using these models, using a proposal distribution based on Poisson superposition. We then use synthetic data, supercomputer event logs, and web search query logs to illustrate that our learning algorithm can efficiently learn nonlinear temporal dependencies, and that our importance sampling algorithm can effectively forecast future events.

## 1 Introduction

The problem of modeling temporal dependencies in temporal streams of discrete events arises in a wide variety of applications. For example, system error logs [14], web search query logs, the firing patterns of neurons [18] and gene expression data [8], can all be viewed as streams of events over time. Events carry both information about their timing and their type (e.g., the web query issued or the type of error logged), and the dependencies between events can be due to both their timing and their types. Modeling these dependencies is valuable for forecasting future events in applications such as system failure prediction for preemptive maintenance or forecasting web users' future interests for targeted advertising.

We introduce the *Piecewise-Constant Conditional Intensity Model* (PCIM), which is a class of marked point processes [4] that can model the types and timing of events. This model captures the dependencies of each type of event on events in the past through a set of piecewise-constant conditional intensity functions. We use decision trees to represent these dependencies and give a conjugate prior for this model, allowing for closed-form computation of the marginal likelihood and parameter posteriors. Model selection then becomes a problem of choosing a decision tree. Decision tree induction can be done efficiently because of the closed form for the marginal likelihood. Forecasting can be carried out using forward sampling for arbitrary finite duration queries. For *episodic sequence queries*, that is, queries that specify particular sequences of events in given future time intervals, we develop a novel approach for estimating the probability of rare queries, which we call the *Poisson Superposition Importance Sampler* (PSIS).

We validate our learning and inference procedures empirically. Using synthetic data we show that PCIMs can correctly learn the underlying dependency structure of event streams, and that the PSIS leads to effective forecasting. We then use real supercomputer event log data to show that PCIMs can be learned more than an order of magnitude faster than Poisson Networks [15, 18], and that they have better test set likelihood. Finally, we show that PCIMs and the PSIS are useful in forecasting future interests of real web search users.

## 2 Related Work

While graphical models such as Bayesian networks [2] and dependency networks [10] are widely used to model the dependencies between variables, they do not model temporal dependencies (see e.g., [8]). Dynamic Bayesian Networks (DBN) [5, 9] allow modeling of temporal dependencies in discrete time. It is not clear how timestamps in our data should be discretized in order to apply the DBN approach. At a minimum, too slow a sampling rate results in poor representation of the data, and too fast a sampling rate increases the number of samples making learning and inference more costly. In addition, allowing long term dependencies requires conditioning on multiple steps into the past, and choosing too fast a sampling rate increases the number of such steps that need to be conditioned on.

Recent progress in modeling continuous time processes include Continuous Time Bayesian Networks (CTBNs) [12, 13], Continuous Time Noisy-Or (CT-NOR) [16], Poisson Cascades [17], and Poisson Networks [15, 18]. CTBNs are homogeneous Markov models of the joint trajectories of discrete finite variables, rather than models of event streams in continuous time [15]. In contrast, CT-NOR and Poisson Cascades model event streams, but require the modeler to choose a parametric form for temporal dependencies. Simma et al [16, 17] describe how this choice significantly impacts model performance, and depends strongly on the domain. In particular, the problem of model selection for CT-NOR and Poisson Cascades is unaddressed. PCIMs, in contrast to CT-NOR and Poisson Cascades, perform structure learning to learn how different events in the past affect future events. Poisson Networks, described in more detail below, are closely related to PCIMs, but PCIMs are over an order of magnitude faster to learn and can model nonlinear temporal dependencies.

## 3 Conditional Intensity Models

In this section, we define Conditional Intensity Models, introduce the class of Piecewise-Constant Conditional Intensity Models, and describe Poisson Networks. We assume that events of different types are distinguished by *labels* $l$ drawn from a finite set $\mathcal{L}$. An *event* is then composed of a non-negative time-stamp $t$ and a label $l$. An *event sequence* $x = \{(t_i, l_i)\}_{i=1}^n$ where $0 < t_1 < \cdots < t_n$. The *history at time* $t$ of event sequence $x$ is the sub-sequence $h(t, x) = \{(t_i, l_i) \mid (t_i, l_i) \in x, t_i \leq t\}$. We write $h_i$ for $h(t_{i-1}, x)$ when it is clear from context which $x$ is meant. By convention $t_0 = 0$. We define the *ending time* $t(x)$ of an event sequence $x$ as the time of the last event in $x$: $t(x) = \max(\{t : (t, l) \in x\})$ so that $t(h_i) = t_{i-1}$.

A *Conditional Intensity Model* (CIM) is a set of non-negative *conditional intensity functions* indexed by label $\{\lambda_l(t|x; \theta)\}_{l \in \mathcal{L}}$. The data likelihood for this model is

$$p(x|\theta) = \prod_{l \in \mathcal{L}} \prod_{i=1}^{n} \lambda_l(t_i|h_i, \theta)^{\mathbf{1}_l(l_i)} e^{-\Lambda_l(t_i|h_i; \theta)} \tag{1}$$

where $\Lambda_l(t|x; \theta) = \int_{-\infty}^{t} \lambda_l(\tau|x; \theta) d\tau$ for each event sequence $x$ and the indicator function $\mathbf{1}_l(l')$ is one if $l' = l$ and zero otherwise. The conditional intensities are assumed to satisfy $\lambda_l(t|x; \theta) = 0$ for $t \leq t(x)$ to ensure that $t_i > t_{i-1} = t(h_i)$. These modeling assumptions are quite weak. In fact, any distribution for $x$ in which the timestamps are continuous random variables can be written in this form. For more details see [4, 6]. Despite the fact that the modeling assumptions are weak, these models offer a powerful approach for decomposing the dependencies of different event types on the past. In particular, this per-label conditional factorization allows one to model detailed label-specific dependence on past events.

### 3.1 Piecewise-Constant Conditional Intensity Models

Piecewise-Constant Conditional Intensity Models (PCIMs) are Conditional Intensity Models where the conditional intensity functions are assumed to be piecewise-constant. As described below, this assumption allows efficient learning and inference. PCIMs are defined in terms of local structures $S_l$ for each label $l$, which specify regions in time where the corresponding conditional intensity function is constant, and local parameters $\theta_l$ for each label which specify the values taken in those regions. *Piecewise-Constant Conditional Intensity Models* (PCIMs) are defined by local structures $S_l = (\Sigma_l, \sigma_l(t, x))$ and local parameters $\theta_l = \{\lambda_{ls}\}_{s \in \Sigma_l}$, where $\Sigma_l$ denotes a set discrete states, $\lambda_{ls}$

are non-negative constants, and $\sigma_l$ denotes a *state function* that maps a time and an event sequence to $\Sigma_l$ and is piecewise constant in time for every event sequence. The conditional intensity functions are defined as $\lambda_l(t|x) = \lambda_{ls}$ with $s = \sigma_l(t, x)$, and thus are piecewise constant. The resulting data likelihood can be written as

$$p(x|S, \theta) = \prod_{l \in \mathcal{L}} \prod_{s \in \Sigma_l} \lambda_{ls}^{c_{ls}(x)} e^{-\lambda_{ls} d_{ls}(x)} \qquad (2)$$

where $S = \{S_l\}_{l \in \mathcal{L}}$, $\theta = \{\theta_l\}_{l \in \mathcal{L}}$, $c_{ls}(x)$ is the number of times label $l$ occurs in $x$ when the state function for $l$ maps to state $s$ (i.e., $\sum_i \mathbf{1}_l(l_i) \mathbf{1}_s(\sigma_l(t_i, h_i))$), and $d_{ls}(x)$ is the total duration during which the state function for $l$ maps to state $s$ in the data $x$ (i.e., $\int_0^{t(x)} \mathbf{1}_s(\sigma(\tau, h(\tau, x))) d\tau$).

### 3.2 Poisson Networks

Poisson networks[15, 18] are closely related to PCIMs. Given a basis set $\mathcal{B}$ of piecewise-constant real-valued feature functions $f(t, x)$, a feature vector $\sigma_l(t, x)$ is defined for each $l$ by selecting component feature functions from $\mathcal{B}$. The resulting $\sigma_l(t, x)$ are piecewise-constant in time. The conditional intensity for $l$ is given by the regression $\lambda_l(t|x, \theta) = e^{w_l \cdot \sigma_l(t, x)}$ with parameter $w_l$. By convention, the component $\sigma_{l,0}(t, x) = 1$ so that $w_{l,0}$ is a bias parameter.

The resulting likelihood does not have a conjugate prior, and in our experiments we use iterative MAP parameter estimates under a Gaussian prior, and use a Laplace approximation of the marginal likelihood for structure learning (i.e., feature selection) [15]. In our experiments, each $f \in \mathcal{B}$ is specified by a label $l$ and a pair of time offsets $0 \le d_1 < d_2$, and takes on the value $\log\left(1 + \frac{c_{l,d_1,d_2}(t, x)}{d_2 - d_1}\right)$ where $c_{l,d_1,d_2}(t, x)$ is the number of times $l$ occurs in $x$ in the interval $[t - d_2, t - d_1)$.

## 4 Learning PCIMs

In this section, we present an efficient learning algorithm for PCIMs. We give a conjugate prior for the parameters $\theta$ which yields closed form formulas for the parameter posteriors and the marginal likelihood of the data given a structure $S$. We then give a decision tree based learning algorithm that uses the closed-form marginal likelihood formula to learn the local structure $S_l$ for each label.

### 4.1 Closed-Form Parameter Posterior and Marginal Likelihood

In general, computing parameter posteriors for likelihoods of the form of equation (1) is complicated. However, in the case of PCIMs, the Gamma distribution is a conjugate prior for $\lambda_{ls}$, despite the fact that the data likelihood of equation (2) is not a product of exponential densities (i.e., when $c_{ls}(x) \ne 1$). The corresponding prior and posterior densities are given by

$$p(\lambda_{ls}|\alpha_{ls}, \beta_{ls}) = \frac{\beta_{ls}^{\alpha_{ls}}}{\Gamma(\alpha_{ls})} \lambda_{ls}^{\alpha_{ls}-1} e^{-\beta_{ls}\lambda_{ls}}; \qquad p(\lambda_{ls}|\alpha_{ls}, \beta_{ls}, x) = p(\lambda_{ls}|\alpha_{ls} + c_{ls}(x), \beta_{ls} + d_{ls}(x))$$

Assuming the prior over $\theta$ is a product of such $p(\lambda_{ls}|\alpha_{ls}, \beta_{ls})$, the marginal likelihood is

$$p(x|S) = \prod_{l \in \mathcal{L}} \prod_{s \in \Sigma_l} \gamma_{ls}(x); \qquad \gamma_{ls}(x) = \frac{\beta_{ls}^{\alpha_{ls}}}{\Gamma(\alpha_{ls})} \frac{\Gamma(\alpha_{ls} + c_{ls}(x))}{(\beta_{ls} + d_{ls}(x))^{\alpha_{ls}+c_{ls}(x)}}$$

In our experiments, we use the point estimate $\hat{\lambda}_{ls} = \frac{\alpha_{ls} + c_{ls}(x)}{\beta_{ls} + d_{ls}(x)}$ which is $\mathbf{E}[\lambda_{ls} \mid x]$.

### 4.2 Structure Learning with Decision Trees

In this section, we specify the set of possible structures in terms of a set of basis state functions, a set of decision trees built from them, and a greedy Bayesian model selection procedure for learning a structure. Finally, we describe the particular set of basis state functions we use in our experiments.

We use $\mathcal{B}$ to denote the set of basis state functions $f(t, x)$, each taking values in a basis state set $\Sigma_f$. Given $\mathcal{B}$, we specify $S_l$ through a decision tree whose interior nodes each have an associated $f \in \mathcal{B}$ and a child corresponding to each value in $\Sigma_f$. The per-label state set $\Sigma_l$ is then the set of

leaves in the tree. The state function $\sigma_l(t, x)$ is computed by recursively applying the basis state functions in the tree until a leaf is reached. Note that the resulting mapping is a valid state function by construction.

In order to carry out Bayesian model selection, we use a factored structural prior $p(S) \propto \prod_{l \in \mathcal{L}} \prod_{s \in \Sigma_l} \kappa_{ls}$. Since the prior and the marginal likelihood both factor over $l$, the local structures $S_l$ can be chosen independently. We search for each $S_l$ as follows. We begin with $S_l$ being the trivial decision tree that maps all event sequences and times to the root. In this case, $\lambda_l(t|x) = \lambda_l$. Given the current $S_l$, we consider $S'_l$ specified by choosing a leaf $s \in \Sigma_l$ and a basis state function $f \in \mathcal{B}$, and assigning $f$ to $s$ to get a set of new child leaves $\{s_1, \cdots, s_m\}$ where $m = |\Sigma_f|$. Because the marginal likelihood factors over states, the gain in the posterior of the structure due to this split is $\frac{p(S'_l|x)}{p(S_l|x)} = \frac{\kappa_{ls_1} \gamma_{ls_1}(x) \cdots \kappa_{ls_m} \gamma_{ls_m}(x)}{\kappa_{ls} \gamma_{ls}(x)}$. The next structure $S'_l$ is chosen by selecting the $s$ and $f$ with the largest gain. The search terminates if there is no gain larger than one. We note that the local structure representation and search can be extended from decision trees to decision graphs in a manner analogous to [3].

In our experiments, we wish to learn how events depend on the timing and type of prior events. We therefore use a set of time and label specific basis state functions. In particular, we use binary basis state functions $f_{l', d_1, d_2, \tau}$ indexed by a label $l' \in \mathcal{L}$, two time offsets $0 \leq d_1 < d_2$ and a threshold $\tau > 0$. Such a $f$ encodes whether or not the event sequence $x$ contains at least $\tau$ events with label $l'$ with timestamps in the window $[t - d_2, t - d_1)$. Examples of decision trees that use such basis state functions are shown in Figure 1.

## 5  Forecasting

In this section, we describe how to use PCIMs to forecast whether a sequence of target labels will occur in a given order and in given time intervals. For example, we may wish to know the probability that a computer system will experience a system failure in the next week and again in the following week, or that an internet user will be shown a particular display ad and then visit the advertising merchant's website in the next month. We call such a sequence and set of associated intervals an *episodic sequence* and denote it by $e = \left\{ \left( l^*_j, [a^*_j, b^*_j) \right) \right\}_{j=1}^k$. We call $(l^*_j, [a^*_j, b^*_j))$ the $j^{th}$ *episode*. We say that the episodic sequence $e$ *occurs* in an event sequence $x$ if $\exists i_1 < \cdots < i_k : (t_{i_j}, l_{i_j}) \in x, l_{i_j} = l^*_j, t_{i_j} \in [a^*_j, b^*_j)$. The set of event sequences $x$ in which $e$ occurs is denoted $\mathcal{X}_e$.

Given an event sequence $h$ and a time $t^* \geq t(h)$, we term any event sequence $x$ whose history up to $t^*$ agrees with $h$ (i.e., $h(t^*, x) = h$) an *extension of $h$ from $t^*$*. Our forecasting problem is, given at observed sequence $h$ at time $t^* \geq t(h)$, to compute the probability that $e$ occurs in extensions of $h$ from $t^*$. This probability is $p(X \in \mathcal{X}_e \mid h(t^*, X) = h)$ and will be denoted using the shorthand $p(\mathcal{X}_e|h, t^*)$. Computing $p(\mathcal{X}_e|h, t^*)$ is hard in general because the probability of episodes of interest can depend on arbitrary numbers of intervening events. We therefore give Monte Carlo estimates for $p(\mathcal{X}_e|h, t^*)$, first describing a forward sampling procedure for forecasting episodic sequences (also applicable to other forecasting problems), and then introducing an importance sampling scheme specifically designed for forecasting episodic sequences.

### 5.1  Forward Sampling

The probability of an episodic sequence can be estimated using a forward sampling approach by sampling $M$ extensions $\{x^{(m)}\}_{m=1}^M$ of $h$ from $t^*$ and using the estimate $\hat{p}_{\text{Fwd}}(\mathcal{X}_e|h, t^*; M) = \frac{1}{M} \sum_{m=1}^M \mathbf{1}_{\mathcal{X}_e}(x^{(m)})$. By Hoeffding's inequality, $P(|\hat{p}_{\text{Fwd}}(\mathcal{X}_e|h, t^*; M) - p(\mathcal{X}_e|h, t^*)| > \epsilon) \leq 2e^{-2\epsilon^2 M}$. Thus, the error in $\hat{p}_{\text{Fwd}}(\mathcal{X}_e|h, t^*; M)$ falls as $O(1/\sqrt{M})$. It is important to note that $\mathbf{1}_{\mathcal{X}_e}(x)$ only depends on $x$ up to $b^*_k$, and thus we need only sample finite extensions $x$ such that $t(x) < b^*_k$ from $p\left(x \mid h(t^*, x) = h, t_{|x|+1} \geq b^*_k\right)$.

The forward sampling algorithm for Poisson Networks [15] can be easily adapted for PCIMs. Here we outline how to forward sample an extension $x$ of $h$ from $t^*$ to $b^*_k$ given a general CIM. Forward sampling consists of iteratively obtaining a sample sequence $x_i$ of length $i$ by sampling $(t_i, l_i)$ and appending to a prior sampled sequence $x_{i-1}$ of length $i - 1$. The CIM likelihood (Equation 1) of an arbitrary event sequence $x$ can be written as $\prod_{i=1}^n p(t_i, l_i|h_i; \theta)$. Thus, we begin with $x_{|h|} = h$,

and iteratively sample $(t_i, l_i)$ from $p(t_i, l_i | h_i = x_{i-1}; \theta)$ and append to $x_{i-1}$ to obtain $x_i$. Note that one needs to use rejection sampling during the first iteration to ensure $t_{|h|+1} > t^*$. The finite extension up to $b_k^*$ is obtained by terminating when $t_i > b_k^*$ and rejecting $t_i$. To sample $(t_i, l_i)$ we note that $p(t_i, l_i | h_i; \theta) = \lambda_{l_i}(t_i | h_i, \theta) e^{-\Lambda_{l_i}(t_i | h_i; \theta)} \prod_{l \neq l_i} e^{-\Lambda_l(t_i | h_i; \theta)}$ has a *competing risks* form [1, 11], so that we can sample $|\mathcal{L}|$ candidate times $t_i^l$ independently from the non-homogeneous exponential densities $\lambda_l(t_i^l | h_i, \theta) e^{-\Lambda_l(t_i^l | h_i; \theta)}$ and then let $t_i$ be the smallest of these candidate times and $l_i$ be the corresponding $l$. A more detailed description of sampling $t_i^l$ from a piecewise constant conditional intensities is given in [15]. Finally, we note that the basic sampling procedure can be made more efficient using the techniques described in [15] and [7].

## 5.2 Importance Sampling

When using a forward sampling approach to forecast unlikely episodic sequences, the episodes of interest will not occur in most of the sampled extensions and our estimate of $p(\mathcal{X}_e | h, t^*)$ will be noisy. In fact, due to the fact that absolute error in $\hat{p}_{\text{Fwd}}$ falls as the square root of the number of sequences sampled, we would need $O(1/p(\mathcal{X}_e | h, t^*)^2)$ sample sequences to get non-trivial lower bounds on $p(\mathcal{X}_e | h, t^*)$ using a forward sampling approach. To mitigate this problem we develop an importance sampling approach, where sequences are drawn from a proposal distribution $q(\cdot)$ that has an increased likelihood of generating extensions in which $\mathcal{X}_e$ occurs, and then uses a weighted empirical estimate. In particular, we will sample extensions $x^{(m)}$ of $h$ from $t^*$ from $q\left(x \mid h(t^*, x) = h, t_{|x|+1} \geq b_k^*\right)$ instead of $p\left(x \mid h(t^*, x) = h, t_{|x|+1} \geq b_k^*\right)$, and will estimate $p(\mathcal{X}_e | h, t^*)$ through

$$\hat{p}_{\text{Imp}}(\mathcal{X}_e | h, t^*; M) = \frac{1}{\sum_{m=1}^M w(x^{(m)})} \sum_{m=1}^M w(x^{(m)}) \mathbf{1}_{\mathcal{X}_e}(x^{(m)}),$$

$$w(x) = \frac{p\left(x \mid h(t^*, x) = h, t_{|x|+1} \geq b_k^*\right)}{q\left(x \mid h(t^*, x) = h, t_{|x|+1} \geq b_k^*\right)}$$

The *Poisson Superposition Importance Sampler* (PSIS) is an importance sampler whose proposal distribution $q$ is based on Poisson superposition. This proposal distribution is defined to be a CIM whose conditional intensity functions are given by $\lambda_l(t | x; \theta) + \lambda_l^*(t | x)$ where $\lambda_l(t | x; \theta)$ is the conditional intensity function of $l$ under the model and $\lambda_l^*(t | x)$ is given by

$$\lambda_l^*(t | x) = \begin{cases} \frac{1}{b_{j(x)}^* - a_{j(x)}(x)} & \text{for } l = l_{j(x)}^*, t \in [a_{j(x)}(x), b_{j(x)}^*), \text{ and } j(x) \neq 0. \\ 0 & \text{otherwise,} \end{cases}$$

where the *active episode* $j(x)$ is 0 if $t(x) \geq b_j(x), j = 1, \cdots, k$ and is $\min(\{j : b_j(x) > t(x)\})$ otherwise. The time $b_j(x)$ when the $j^{\text{th}}$ episode ceases to be active is the time at which the $j^{\text{th}}$ episode occurs in $x$, or $b_j^*$ if it does not occur. If the episodic intervals $[a_j^*, b_j^*]$ do not overlap, $a_j(x) = a_j^*$. In general $a_j(x)$ and $b_j(x)$ are given by the recursion

$$a_j(x) = \max\left(\{a_j^*, b_{j-1}(x)\}\right)$$
$$b_j(x) = \min\left(\{b_j^*\} \cup \{(t_i, l_i) \in x : l_i = l_j^*, t_i \in [a_j(x), b_j^*)\}\right).$$

This choice of $q$ makes it likely that the $j^{\text{th}}$ episode will occur after the $j - 1^{\text{th}}$ episode.

As the proposal distribution is also a CIM, importance sampling can be done using the forward sampling procedure above. If the model is a PCIM, the proposal distribution is also a PCIM, since $\lambda_l^*(t | x)$ are piecewise constant in $t$. In practice the computation of $j(x)$, $a_j(x)$, and $b_j(x)$ can be done during forward sampling.

The importance weight corresponding to our proposal distribution is

$$w(x) = \prod_{j=1}^k \exp\left(\frac{b_j(x) - a_j(x)}{b_j^* - a_j(x)}\right) \prod_{\substack{(t_i, l_i) \in x: \\ t_i = b_j(x), l_i = l_j^*}} \frac{\lambda_{l_j^*}(t_i | x_i)}{\lambda_{l_j^*}(t_i | x_i) + \frac{1}{b_j^* - a_j(x)}}.$$

In many problems, the importance weight $w(x)$ of a sequence $x$ of length $n$ is a product of $n$ small terms. When $n$ large, this can cause the importance weights to become degenerate, and this problem is often solved using particle filtering [7]. Note that the second product in $w(x)$ above has at most one term for each $j$ so that $w(x)$ has $k$ terms corresponding to the $k$ episodes, which is independent of $n$. Thus, we do not experience the problem of degenerate weights when $k$ is small, regardless of the number of events sampled.

## 6 Experimental Results

We first validate that PCIMs can learn temporal dependencies and that the PSIS gives faster forecasting than forward sampling using a synthetic data set. We then show that PCIMs are more than an order of magnitude faster to train than Poisson Networks, and better model unseen test data using real supercomputer log data. Finally we show that PCIMs and the PSIS allow the forecasting future interests of web search users using real log data from a major commercial search engine.

### 6.1 Validation on Synthetic Data

In order to evaluate the ability of PCIMs to learn nonlinear temporal dependencies we sampled data from a known model and verified that the dependencies learned were correct. Data was sampled from a PCIM with $\mathcal{L} = \{\text{A, B, C}\}$. The known model is shown in Figure 1.

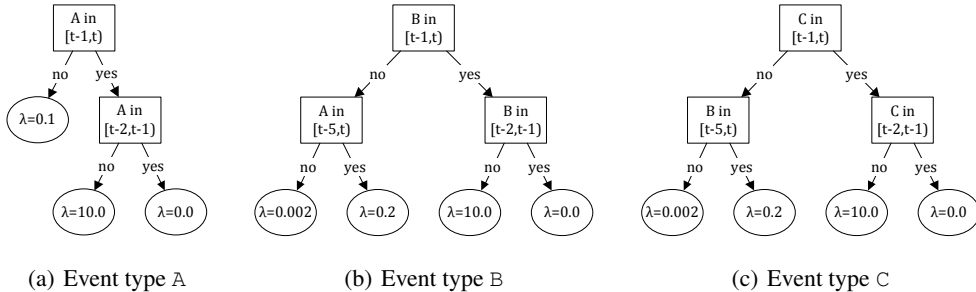

(a) Event type A        (b) Event type B        (c) Event type C

Figure 1: Decision trees representing $S$ and $\theta$ for events of type A, B and C.

We sampled 100 time units of data, observing 97 instances of A, 58 instances of B, and 71 instances of C. We then learn a PCIM from the sampled data. We used basis state functions that tested for the presence of each label in windows with boundaries at $t - 0, 1, 2, \cdots, 10$, and $+\infty$ time units. We used a common prior with a mean rate of $0.1$ and a equivalent sample size of one time unit for all $\lambda_{ls}$, and the structural prior described above with $\kappa_{ls} = 0.1$ for all $s$.

The learned PCIM perfectly recovered the correct model structure. We repeated the experiment by sampling data from a model with fifteen labels, consisting of five independent copies of the model above. That is, $\mathcal{L} = \{\text{A}_1, \text{B}_1, \text{C}_1, \cdots, \text{A}_5, \text{B}_5, \text{C}_5\}$ with each triple $\text{A}_i, \text{B}_i, \text{C}_i$ independent of other labels, and dependent on each other as specified by Figure 1. Once again, the model structure was recovered perfectly.

We evaluated the PSIS in forecasting event sequences with the model shown in Figure 1. The convergence of importance sampling is compared with that of forward sampling in Figure 2. We give results for forecasting three different episodic sequences, consisting of the label sequences $\{\text{C}\}$, $\{\text{C, B}\}$, and $\{\text{C, B, A}\}$, all in the interval $[0, 1]$, given an empty history. The three queries are given in order of decreasing probability, so that inference becomes harder. We show how estimates of the probabilities of given episodic sequences vary as a function of the number of sequences sampled, giving the mean and variance of the trajectories of the estimates computed over ten runs. For all three queries, importance sampling converges faster and has lower variance. Since exact inference is infeasible for this model, we forward sample 4,000,000 event sequences and display this estimate. Note that despite the large sample size the Hoeffding bound gives a 95% confidence

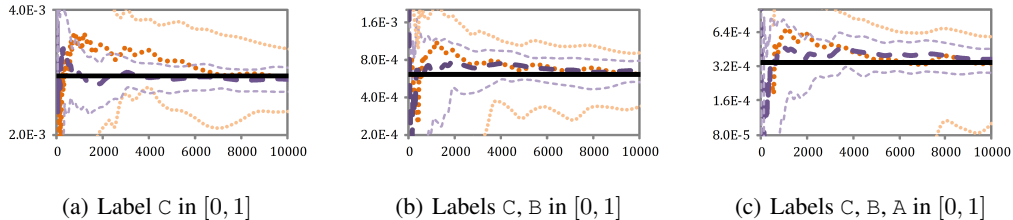

|  (a) Label C in $[0, 1]$ | (b) Labels C, B in $[0, 1]$ | (c) Labels C, B, A in $[0, 1]$ |

Figure 2: Trajectories of $\hat{p}_{\text{Imp}}$ and $\hat{p}_{\text{Fwd}}$ vs. the number of sequences sampled for three different queries. The dashed and dotted lines show the empirical mean and standard deviation over ten runs of $\hat{p}_{\text{Imp}}$ and $\hat{p}_{\text{Fwd}}$. The solid line shows $\hat{p}_{\text{Fwd}}$ based on 4 million event sequences.

interval of $\pm 0.0006$ for this estimate, which is large relative to the probabilities estimated. This further suggests the need for importance sampling for rare label sequences.

## 6.2 Modeling Supercomputer Event Logs

We compared PCIM and Poisson Nets on the task of modeling system event logs from the Blue-Gene/L supercomputer at Lawrence Livermore National Laboratory [14], available at the USENIX Computer Failure Data Repository. We filtered out informational (non-alert) messages from the logs, and randomly split the events by node into a training set with 311,060 alerts from 21,962 nodes, and a test set with 68,502 alerts from 9,412 nodes. We learned dependencies between the 38 alert types in the data. We treat the events from each node as separate sequences, and use a product of the per-sequence likelihoods given in equation (1).

For both models, we used window boundaries at $t - 1/60, 1, 60, 3600,$ and $\infty$ seconds. The PCIM used count threshold basis state functions with thresholds of $1, 4, 16$ and $64$ while the Poisson Net used log count feature vectors as described above. Both models used priors with a mean rate of an event every 100 days, no dependencies, and an equivalent sample size of one second. Both used a structural prior with $\kappa_{ls} = 0.1$. Table 1 shows the test set likelihood and the run time for the two approaches. PCIM achieves better test set likelihood and is more than an order of magnitude faster.

|  | Test Log Likelihood | Training Time |
|---|---|---|
| PCIM | -85.3 | 11 min |
| Poisson Net | -88.8 | 3 hr 33 min |

Table 1: A comparison of the PCIM and Poisson Net in modeling supercomputer event logs. The test set log likelihood reported has been divided by the number of test nodes (9,412). The training time for the PCIM and Poisson Net are also shown.

## 6.3 Forecasting Future Interests of Web Search Users

We used the query logs of a major internet search engine to investigate the use of PCIMs in forecasting the future interests of web search users. All queries are mapped to one of 36 different interest categories using an automatic classifier. Thus, $\mathcal{L}$ contains 36 labels, such as "Travel" or "Health & Wellness." Our training set contains event sequences for approximately 23k users consisting of about 385k timestamped labels recorded over a two month period. The test set contains event sequences for approximately 11k users of about 160k timestamped labels recorded over the next month.

We trained a PCIM on the training data using window boundaries at $t - 1$ hour, $t - 1$ day, and $t - 1$ week, and basis state functions that tested for the presence of one or more instance of each label in each window, treating users as i.i.d. The prior had a mean rate of an event every year, an equivalent sample size of one day. The structural prior had $\kappa_{ls} = 0.1$. The model took 1 day and 18 hours to train on 3 GHz workstation. We did not compare to a Poisson network on this data since, as shown above, Poisson networks take an order of magnitude longer to learn.

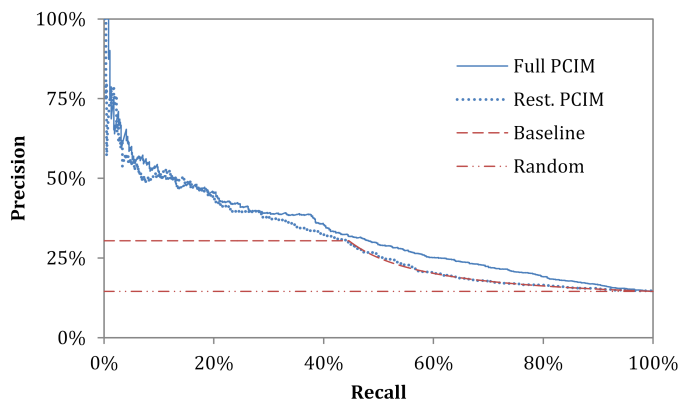

Figure 3: Precision-recall curves for forecasting future Health & Wellness queries using a full PCIM, a restricted PCIM that conditions only on past Health & Wellness queries, a baseline that takes into account only past Health & Wellness queries and not their timing, and random guessing.

Given the first week of each test user's event sequence, we forecasted whether they would issue a query in a chosen target category in the second week. We used the PSIS with 100 sample sequences for forecasting. Figure 3 shows the precision recall curve for one target category label. Also shown is the result for restricted PCIMs that only model dependencies on prior occurrences of the target category. This is compared to a baseline where the conditional intensity depends only on whether the target label appeared in the history. This shows that modeling the temporal aspect of dependencies does provide a large improvement. Modeling dependencies on past occurrences of other labels also provides an improvement in the right-hand region of the precision-recall curve.

To better understand the performance of PCIMs we also examined the problem of predicting the first occurrence of the target label. As Figure 3 suggests (but doesn't show), the PCIM can model cross-label dependencies to forecast the first occurrence of the target label. Forecasting new interests is valuable in a variety of applications including advertising and the fact that PCIMs are able to forecast first occurrences is promising. Results similar to Figure 3 were obtained for other target labels.

## 7 Discussion

We presented the Piecewise-Constant Conditional Intensity Model, which is a model of temporal dependencies in continuous time event streams. We gave a conjugate prior and a greedy tree building procedure that allow for efficient learning of these models. Dependencies on the history are represented through automatically learned combinations of a given set of basis state functions. One of the key benefits of PCIMs is that they allow domain knowledge to be encoded in these basis state functions. This domain knowledge is incorporated into the model during structure search in situations where it is supported by the data. The fact that we use decision trees allows us to easily interpret the learned dependencies.

In this paper, we focused on basis state functions indexed by a fixed set of time windows and labels. Exploring alternative types of basis state functions is an area for future research. For example, basis state functions could encode the most recent events that have occurred in the history rather than the events that occurred in windows of interest. The capacity of the resulting model class depends on the set of basis state functions chosen. Understanding how to choose the basis state functions and how to adapt our learning procedure to control the resulting capacity is another open topic. We also presented the Poisson Superposition Importance Sampler for forecasting episodic sequences with PCIMs. Developing forecasting algorithms for more general queries is of interest.

Finally, we demonstrated the value of PCIMs in modeling the temporal behavior of web search users and of supercomputer nodes. In many applications, we have access to richer event streams such as spatio-temporal event streams and event streams with structured labels. It would be interesting to extend PCIMs to handle such rich event streams.

# References

[1] Simeon M. Berman. Note on extreme values, competing risks and semi-Markov processes. *Ann. Math. Stat.*, 34(3):1104–1106, 1963.

[2] W. Buntine. Theory refinement on Bayesian networks. In *UAI*, 1991.

[3] David Maxwell Chickering, David Heckerman, and Christopher Meek. A Bayesian approach to learning Bayesian networks with local structure. In *UAI*, 1997.

[4] D. J. Daley and D. Vere-Jones. *An Introduction to the Theory of Point Processes: Elementary Theory and Methods*, volume I. Springer, 2 edition, 2003.

[5] Thomas Dean and Keiji Kanazawa. Probabilistic temporal reasoning. In *AAAI*, 1988.

[6] Vanessa Didelez. Graphical models for marked point processes based on local independence. *J. Roy. Stat. Soc., Ser. B*, 70(1):245–264, 2008.

[7] Yu Fan and Christian R. Shelton. Sampling for approximate inference in continuous time Bayesian networks. In *AI & M*, 2008.

[8] N. Friedman, I. Nachman, and D. Peér. Using Bayesian networks to analyze expression data. *J. Comp. Bio.*, 7:601–620, 2000.

[9] Nir Friedman, Kevin Murphy, and Stuart Russell. Learning the structure of dynamic probabilistic networks. In *UAI*, 1998.

[10] David Heckerman, David Maxwell Chickering, Christopher Meek, Robert Rounthwaite, and Carl Kadie. Dependency networks for inference, collaborative filtering, and data visualization. *JMLR*, 1:49–75, October 2000.

[11] A. A. J. Marley and Hans Colonius. The "horse race" random utility model for choice probabilities and reaction times, and its competing risks interpretation. *J. Math. Psych.*, 36:1–20, 1992.

[12] Uri Nodelman, Christian R. Shelton, and Daphne Koller. Continuous time Bayesian networks. In *UAI*, 2002.

[13] Uri Nodelman, Christian R. Shelton, and Daphne Koller. Expectation Maximization and complex duration distributions for continuous time Bayesian networks. In *UAI*, 2005.

[14] Adam Oliner and Jon Stearley. What supercomputers say - an analysis of five system logs. In *IEEE/IFIP Conf. Dep. Sys. Net.*, 2007.

[15] Shyamsundar Rajaram, Thore Graepel, and Ralf Herbrich. Poisson-networks: A model for structured point processes. In *AIStats*, 2005.

[16] Aleksandr Simma, Moises Goldszmidt, John MacCormick, Paul Barham, Richard Brock, Rebecca Isaacs, and Reichard Mortier. CT-NOR: Representing and reasoning about events in continuous time. In *UAI*, 2008.

[17] Aleksandr Simma and Michael I. Jordan. Modeling events with cascades of Poisson processes. In *UAI*, 2010.

[18] Wilson Truccolo, Uri T. Eden, Matthew R. Gellows, John P. Donoghue, and Emery N. Brown. A point process framework relating neural spiking activity to spiking history, neural ensemble, and extrinsic covariate effects. *J. Neurophysiol.*, 93:1074–1089, 2005.

